# A Lagrangian Formulation For Optical Backpropagation Training In Kerr-Type Optical Networks

**James E. Steck**
Mechanical Engineering
Wichita State University
Wichita, KS 67260-0035

**Steven R. Skinner**
Electrical Engineering
Wichita State University
Wichita, KS 67260-0044

**Alvaro A. Cruz-Cabrara**
Electrical Engineering
Wichita State University
Wichita, KS 67260-0044

**Elizabeth C. Behrman**
Physics Department
Wichita State University
Wichita, KS 67260-0032

## Abstract

A training method based on a form of *continuous* spatially distributed optical error back-propagation is presented for an *all optical network* composed of *nondiscrete* neurons and weighted interconnections. The all optical network is feed-forward and is composed of thin layers of a Kerr-type self focusing/defocusing nonlinear optical material. The training method is derived from a Lagrangian formulation of the constrained minimization of the network error at the output. This leads to a formulation that describes training as a calculation of the distributed error of the optical signal at the output which is then reflected back through the device to assign a spatially distributed error to the internal layers. This error is then used to modify the internal weighting values. Results from several computer simulations of the training are presented, and a simple optical table demonstration of the network is discussed.

# 1   KERR TYPE MATERIALS

Kerr-type optical networks utilize thin layers of Kerr-type nonlinear materials, in which the index of refraction can vary within the material and depends on the amount of light striking the material at a given location. The material index of refraction can be described by: $n(x)=n_0+n_2I(x)$, where $n_0$ is the linear index of refraction, $n_2$ is the nonlinear coefficient, and $I(x)$ is the irradiance of a applied optical field as a function of position x across the material layer (Armstrong, 1962). This means that a beam of light (a signal beam carrying information perhaps) passing through a layer of Kerr-type material can be steered or controlled by another beam of light which applies a spatially varying pattern of intensity onto the Kerr-type material. Steering of light with a glass lens (having constance index of refraction) is done by varying the thickness of the lens (the amount of material present) as a function of position. Thus the Kerr effect can be **loosely** thought of as a glass lens whose geometry and therefore focusing ability could be dynamically controlled as a function of position across the lens. Steering in the Kerr material is accomplished by a gradient or change in the material index of refraction which is created by a gradient in applied light intensity. This is illustrated by the simple experiment in Figure 1 where a small weak probe beam is steered away from a straight path by the intensity gradient of a more powerful pump beam.

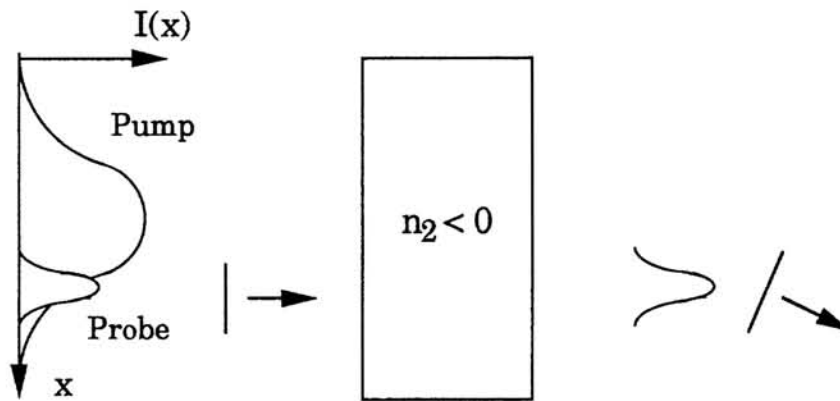

**Figure 1**: Light Steering In Kerr Materials

# 2   OPTICAL NETWORKS USING KERR MATERIALS

The Kerr optical network, shown in Figure 2, is made up of thin layers of the Kerr- type nonlinear medium separated by thick layers of a linear medium (free space) (Skinner, 1995). The signal beam to be processed propagates optically in a direction z perpendicular to the layers, from an input layer through several alternating linear and nonlinear layers to an output layer. The Kerr material layers perform the nonlinear processing and the linear layers serve as connection layers. The input ($I(x)$) and the weights ($W_1(x), W_2(x) ... W_n(x)$) are irradiance fields applied to the Kerr type layers, as functions of lateral position x, thus varying the

refractive index profile of the nonlinear medium. Basically, the applied weight irradiences steer the signal beam via the Kerr effect discussed above to produce the correct output. The advantage of this type of optical network is that both neuron processing and weighted connections are achieved by uniform layers of the Kerr material. The all optical nature eliminates the need to physically construct neurons and connections on an individual basis.

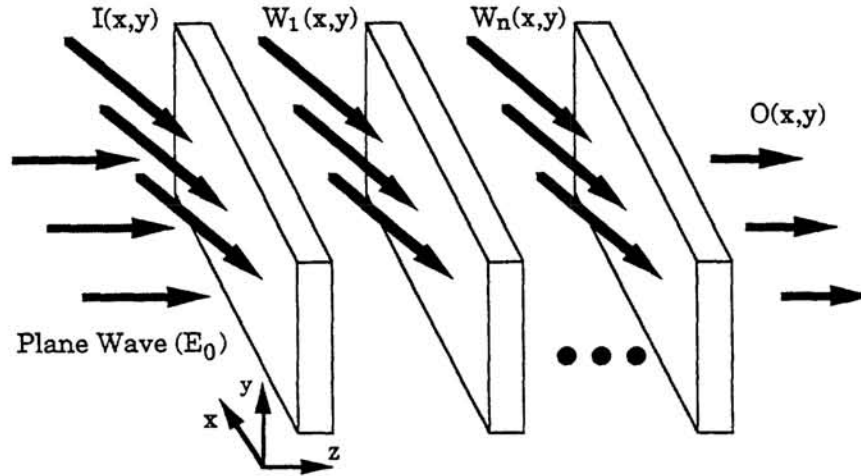

**Figure 2**: Kerr Optical Neural Network Architecture

If $E_i(\alpha)$ is the light entering the $i^{th}$ nonlinear layer at lateral position $\alpha$, then the effect of the nonlinear layer is given by

$$F_i(\alpha) = E_i(\alpha) \; e^{-jk_0 \Delta NL_i n_2 (|W_i(\alpha)|^2 + |E_i(\alpha)|^2)} \qquad (1)$$

where $W_i(\alpha)$ is the applied weight field. Transmission of light at lateral location $\alpha$ at the beginning of the $i^{th}$ linear layer to location $\beta$ just before the $i+1^{th}$ nonlinear layer is given by

$$E_{i+1}(\beta) = \frac{jC_i}{\pi} \int_{\alpha_i} F_i(\alpha) \; e^{jC_i(\beta - \alpha)^2} d\alpha \qquad where \qquad C_i = \frac{k_0}{2 \Delta L_i} \qquad (2)$$

## 3  OPTICAL BACK-PROPAGATION TRAINING

Traditional feed-forward artificial neural networks composed of a *finite* number of *discrete* neurons and weighted connections can be trained by many techniques. Some of the most successful techniques are based upon the well known training method called back-propagation which results from minimizing the network output error, with respect to the network weights by a gradient descent algorithm. The optical network is trained using a form of *continuous* optical back-propagation which is developed for a *nondiscrete* network. Gradient descent is applied to minimize the error over the entire output region of the optical network. This error is a continuous distribution of error calculated over the output region.

Optical back-propagation is a specific technique by which this error distribution is **optically** propagated backward through the linear and nonlinear optical layers to produce error signals by which the light applied to the nonlinear layers is modified. Recall that this applied light $W_i$ controls what serves as connection "weights" in the optical network. Optical back-propagation minimizes the error $L_0$ over an output region $\Omega_0$, a subdomain of the final or $n^{th}$ layer of the network,

$$L_0 = \frac{1}{2\gamma^2}[\ I_0 - D\ ]^2 \qquad where \qquad I_0 = \gamma \int_{\Omega_0} O(\alpha) O^*(\alpha) d\alpha \qquad (3)$$

subject to the constraint that the propagated light, $E_i(\alpha)$, satisfies the equations of forward propagation (1) and (2). $O(\beta) = E_{n+1}(\beta)$ and is the network output, $\gamma$ is a scaling factor on the output intensity. $L_0$ then is the squared error between the desired output value D and the average intensity $I_0$ of the output distribution $O(\beta)$.

This constrained minimization problem is posed in a Lagrange formulation similar to the work of (le Cunn, 1988) for conventional feedforward networks and (Pineda, 1987) for conventional recurrent networks; the difference being that for the optical network of this paper the Electric field E and the Lagrange multiplier are complex and also continuous in the spatial variable thus requiring the Lagrangian below. A Lagrangian is defined as;

$$L = I_0 + \sum_{i=1}^{n} \int_{\Omega_i} \lambda_i^*(\alpha)\ [\ E_{i+1}(\alpha)\ -\ \int_{\Omega_i} F_i(\beta)\frac{jC_i}{\pi}e^{-jC_i(\beta-\alpha)^2}\ d\beta\ ]\ d\alpha$$

$$+ \sum_{i=1}^{n} \int_{\Omega_i} \lambda_{i+1}(\alpha)\ [\ E_{i+1}(\alpha)\ -\ \int_{\Omega_i} F_i(\beta)\frac{jC_i}{\pi}e^{-jC_i(\beta-\alpha)^2}\ d\beta\ ]^*\ d\alpha \qquad (4)$$

Taking the variation of L with respect to $E_i$, the Lagrange multipliers $\lambda_i$, and using gradient descent to minimize L with respect to the applied weight fields $W_i$ gives a set of equations that amount to calculating the error at the output and propagating the error optically backwards through the network. The pertinent results are given below. The distributed assignment of error on the output field is calculated by

$$\lambda_{n+1}(\beta) = \frac{1}{\gamma} O^*(\beta)[\ D - I_0\ ] \qquad (5)$$

This error is then propagated back through the $n^{th}$ or final *linear* optical layer by the equation

$$\delta_n(\beta) = \frac{jC_n}{\pi} \int_{\Omega_0} \lambda_{n+1}(\alpha)\ e^{-jC_n(\beta-\alpha)^2} d\alpha \qquad (6)$$

which is used to update the "weight" light applied to the $n^{th}$ nonlinear layer. Optical back-propagation, through the $i^{th}$ *nonlinear* layer (giving $\lambda_i(\beta)$) followed by the *linear* layer (giving $\delta_{i-1}(\beta)$) is performed according to the equations

$$\lambda_i(\beta) = \delta_i(\beta) \ e^{j k_0 \Delta N L_i n_2 (|W_i(\beta)|^2 + |E_i(\beta)|^2)}$$

$$+ \ k_0 \Delta N L n_2 E_i^*(\beta) \ 2 \ IM[ \ \delta_i(\beta) \ E_i(\beta) \ e^{j k_0 \Delta N L_i n_2 (|W_i(\beta)|^2 + |E_i \beta|^2)} \ ] \tag{7}$$

$$\delta_{i-1}(\beta) = \frac{j C_{i-1}}{\pi} \int_{Q_i} \lambda_i(\alpha) \ e^{-\mathcal{L}_{i-1}(\beta - \alpha)^2} d\alpha$$

This gives the error signal $\delta_{i-1}(\beta)$ used to update the "weight" light distribution $W_{i-1}(\beta)$ applied to the i-1 nonlinear layer. The "weights" are updated based upon these errors according to the gradient descent rule

$$W_i^{new}(\beta) = W_i^{old}(\beta)$$

$$+\eta_i(\beta) k_0 \Delta N L_i n_2 W_i^{old}(\beta) \ 2 \ IM [ \ E_i(\beta) \ \delta_i(\beta) \ e^{-k_0 \Delta N L_i n_2 (|W_i^{old}(\beta)|^2 + |E_i(\beta)|^2)} \ ] \tag{8}$$

where $\eta_i(\beta)$ is a learning rate which can be, but usually is not a function of layer number i and spatial position $\beta$. Figure 3 shows the optical network (thick linear layers and thin nonlinear layers) with the uniform plane wave $E_0$, the input signal distribution I, forward propagation signals $E_1 \ E_2 \dots E_n$, the weighting light distributions at the nonlinear layers $W_1 \ W_2 \dots W_n$. Also shown are the error signal $\lambda_{n+1}$ at the output and the back-propagated error signals $\delta_n \dots \delta_2 \ \delta_1$ for updating the nonlinear layers. Common nonlinear materials exist for which the material constants are such that the second term in the first of Equations 7 becomes small. Ignoring this second term gives an approximate form of optical back-propagation which amounts to **calculating the error at the output of the network and then reversing its direction to optically propagate this error backward through the device**. This can be easily seen by comparing Equations 6 and 7 (with the second term dropped) for optical back-propagation of the output error $\lambda_n$ with Equations 1 and 2 for the forward propagation of the signal $E_i$. This means that the optical back-propagation training calculations potentially can be implemented in the same physical device as the forward network calculations. Equation (8) then becomes;

$$W_i^{new}(\beta) = W_i^{old}(\beta)$$

$$+ (2\eta_i(\beta) k_0 \Delta N L_i n_2) \ W_i^{old}(\beta) [ \ (E_i(\beta) \ \lambda_i(\beta)) - (E_i(\beta) \ \lambda_i(\beta))^* \ ] \tag{9}$$

which **may** be able to be implemented optically.

## 4   SIMULATION RESULTS

To prove feasibility, the network was then trained and tested on several benchmark classification problems, two of which are discussed here. More details on these and other simulations of the optical network can be found in (Skinner, 1995). In the first (Using Nworks, 1991), iris species were classified into one of three categories: Setosa, Versicolor or Virginica. Classification was based upon length and width of the sepals and

petals. The network consisted of an input self-defocusing layer with an applied irradiance field which was divided into 4 separate Gaussian distributed input regions 25 microns in width followed by a linear layer. This pattern is repeated for 4 more groups composed of a nonlinear layer (with applied weights) followed by a linear layer. The final linear layer has three separate output regions 10 microns wide for binary classification as to species. The nonlinear layers were all 20 microns thick with $n_2 = -.05$ and the linear layers were 100 microns thick. The wavelength of applied light was 1 micron and the width of the network was 512 microns discretized into 512 pixels. This network was trained on a set of 50 training pairs to produce correct classification of all 50 training pairs. The network was then used to classify 50 additional pairs of test data which were not used in the training phase. The network classified 46 of these correctly for a 92% accuracy level which is comparable to a standard feedforward network with discrete sigmoidal neurons.

In the second problem, we tested the performance of the network on a set of data from a dolphin sonar discrimination experiment (Roitblat, 1991). In this study a dolphin was presented with one of three different types of objects (a tube, a sphere, and a cone), allowed to echolocate, and rewarded for choosing the correct one from a comparison array. The Fourier transforms of his click echoes, in the form of average amplitudes in each of 30 frequency bins, were then used as inputs for a neural network. Nine nonlinear layers were used along with 30 input regions and 3

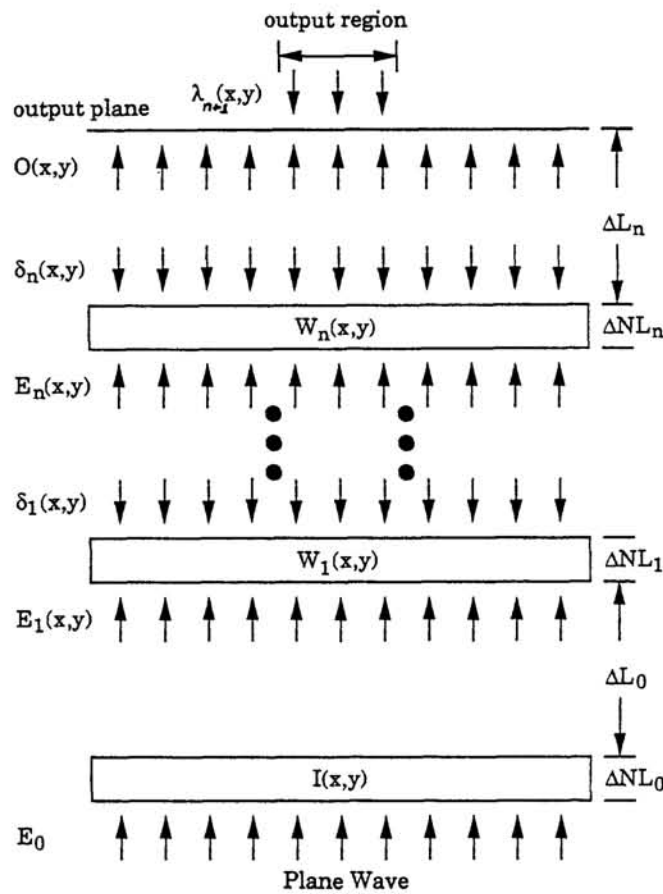

**Figure 3**: Optical Network Forward Data and Backward Error Data Flow

output regions, the remainder of the network physical parameters were the same as above for the iris classification. Half the data (13 sets of clicks) was used to train the network, with the other half of the data (14 sets) used to test the training. After training, classification of the test data set was 100% correct.

## 5  EXPERIMENTAL RESULTS

As a proof of the concept, the optical neural network was constructed in the laboratory to be trained to perform various logic functions. Two thermal self-defocusing layers were used, one for the input and the other for a single layer of weighting. The nonlinear coefficient of the index of refraction ($n_2$) was measured to be $-3 \times 10^{-4}$ $cm^2/W$. The nonlinear layers had a thickness ($\Delta NL_0$ and $\Delta NL_1$) of 630$\mu$m and were separated by a distance ($\Delta L_0$) of 15cm. The output region was 100$\mu$m wide and placed 15cm ($\Delta L_1$) behind the weighting layer. The experiment used HeNe laser light to provide the input plane wave and the input and weighting irradiances. The spatial profiles of the input and weighting layers were realized by imaging a LCD spatial light modulator onto the respective nonlinear layers. The inputs were two bright or dark regions on a Gaussian input beam producing the intensity profile:

$$I(x) = I_0\, e^{-(t/K_0)^2}[1 + Q_o\, rect((x + x_0)/K_1)][1 + Q_1 rect((x - x_0)/K_1)]$$

where $I_0 = 12.5$ mW/$cm^2$, $K_0 = 900\mu$m, $x_0 = 600\mu$m, $K_1 = 400\mu$m, and $Q_0$ and $Q_1$ are the logic inputs taking on a value of zero or one. The weight profile $W_1(x) = I_0 exp[-(x/K_0)^2][1 + w_1(x)]$ where $w_1(x)$ can range from zero to one and is found through training using an algorithm which probed the weighting mask in order update the training weights. Table 1 shows the experimental results for three different logic gates. Given is the normalized output before and after training. The network was trained to recognize a logic zero for a normalized output $\leq 0.9$ and a logic one or a normalized output $\geq 1.1$. An output value greater than 1 is considered a logic one and an output value less than one is a logic zero. RME is the root mean error.

## 6  CONCLUSIONS

Work is in progress to improve the logic gate results by increasing the power of propagating signal beam as well as both the input and weighting beams. This will effectively increase the nonlinear processing capability of the network since a higher power produces more nonlinear effect. Also, more power will allow expansion of all of the beams thereby increasing the effective resolution of the thermal materials. This reduces the effect of heat transfer within the material which tends to wash out or diffuse benificial steep gradients in temperature which are what produce the gradients in the index of refraction. In addition, the use of photorefractive crystals for optical weight storage shows promise for being able to optically phase conjugate and backpropagate the output error as well as implement the weight update rule for all optical network training. This appears to be simpler than optical networks using volume hologram weight storage because the Kerr network requires only planar hologram storage.

| | Inputs | 0  0 | 0  1 | 1  0 | 1  1 | RME |
|---|---|---|---|---|---|---|
| **NOR** | Start | 1.001 | .802 | .698 | .807 | 7.3% |
| | Finish | 1.110 | .884 | .772 | .896 | 0 |
| | Change | .109 | .082 | .074 | .089 | -7.3% |
| | Output | 1 | 0 | 0 | 0 | |
| **AND** | Start | .998 | 1.092 | 1.148 | 1.440 | 16.4% |
| | Finish | .757 | .855 | .894 | 1.124 | 0 |
| | Change | -.241 | -.237 | -.254 | -.316 | -16.4% |
| | Output | 0 | 0 | 0 | 1 | |
| **XNOR** | Start | .998 | .880 | .893 | .994 | 7.3% |
| | Finish | 1.084 | .933 | .928 | 1.073 | 2.7% |
| | Change | .086 | .053 | .035 | .079 | -4.6% |
| | Output | 1 | 0 | 0 | 1 | |

**Table 1**: Preliminary Experimental Logic Gate Results

## References

Armstrong, J.A., Bloembergen, N., Ducuing, J., and Pershan, P.S., (1962) "Interactions Between Light Waves in a Nonlinear Dielectric", *Physical Review*, Vol. 127, pp. 1918-1939.

le Cun, Yann, (1988) "A Theoretical Framework for Back-Propagation", Proceedings of the 1988 Connectionist Models Summer School, Morgan Kaufmann, pp. 21-28.

Pineda, F.J., (1987) "Generalization of backpropagation to recurrent and higher order neural networks", Proceedings of IEEE Conference on Neural information Processing Systems, November 1987, IEEE Press.

Roitblat., Moore, Nachtigall, and Penner, (1991) "Natural dolphin echo recognition using an integrator gateway network," in *Advances in Neural Processing Systems 3* Morgan Kaufmann, San Mateo, CA, 273-281.

Skinner, S.R., Steck, J.E., Behrman, E.C., (1995) "An Optical Neural Network Using Kerr Type Nonlinear Materials", To Appear in *Applied Optics*.

"Using Nworks, (1991) *An Extended Tutorial for NeuralWorks Professional II/Plus and NeuralWorks Explorer*, NeuralWare, Inc. Pittsburgh, PA, pg. UN-18.